# Pranking with Ranking

**Koby Crammer**   and   **Yoram Singer**
School of Computer Science & Engineering
The Hebrew University, Jerusalem 91904, Israel
{kobics,singer}@cs.huji.ac.il

## Abstract

We discuss the problem of ranking instances. In our framework each instance is associated with a rank or a rating, which is an integer from 1 to $k$. Our goal is to find a rank-prediction rule that assigns each instance a rank which is as close as possible to the instance's true rank. We describe a simple and efficient online algorithm, analyze its performance in the mistake bound model, and prove its correctness. We describe two sets of experiments, with synthetic data and with the EachMovie dataset for collaborative filtering. In the experiments we performed, our algorithm outperforms online algorithms for regression and classification applied to ranking.

## 1  Introduction

The ranking problem we discuss in this paper shares common properties with both classification and regression problems. As in classification problems the goal is to assign one of $k$ possible labels to a new instance. Similar to regression problems, the set of $k$ labels is structured as there is a total order relation between the labels. We refer to the labels as ranks and without loss of generality assume that the ranks constitute the set $\{1, 2, \ldots, k\}$. Settings in which it is natural to rank or rate instances rather than classify are common in tasks such as information retrieval and collaborative filtering. We use the latter as our running example. In collaborative filtering the goal is to predict a user's rating on new items such as books or movies given the user's past ratings of the similar items. The goal is to determine whether a movie fan will like a new movie and to what degree, which is expressed as a rank. An example for possible ratings might be, `run-to-see`, `very-good`, `good`, `only-if-you-must`, and `do-not-bother`. While the different ratings carry meaningful semantics, from a learning-theoretic point of view we model the ratings as a totally ordered set (whose size is 5 in the example above).

The interest in ordering or ranking of objects is by no means new and is still the source of ongoing research in many fields such mathematical economics, social science, and computer science. Due to lack of space we clearly cannot cover thoroughly previous work related to ranking. For a short overview from a learning-theoretic point of view see [1] and the references therein. One of the main results of [1] underscores a complexity gap between classification learning and ranking learning. To sidestep the inherent intractability problems of ranking learning several approaches have been suggested. One possible approach is to cast a ranking problem as a regression problem. Another approach is to reduce a total order into a set of pref-

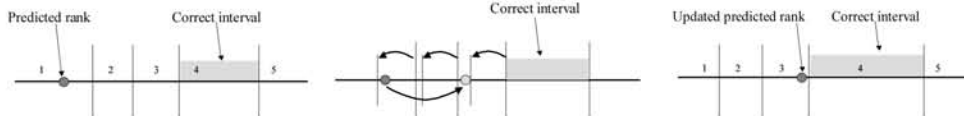

Figure 1: An Illustration of the update rule.

erences over pairs [3, 5]. The first case imposes a metric on the set of ranking rules which might not be realistic, while the second approach is time consuming since it requires increasing the sample size from $n$ to $O(n^2)$.

In this paper we consider an alternative approach that directly maintains a totally ordered set via projections. Our starting point is similar to that of Herbrich et. al [5] in the sense that we project each instance into the reals. However, our work then deviates and operates directly on rankings by associating each ranking with distinct sub-interval of the reals and adapting the support of each sub-interval *while* learning. In the next section we describe a simple and efficient online algorithm that manipulates concurrently the direction onto which we project the instances and the division into sub-intervals. In Sec. 3 we prove the correctness of the algorithm and analyze its performance in the mistake bound model. We describe in Sec. 4 experiments that compare the algorithm to online algorithms for classification and regression applied to ranking which demonstrate the merits of our approach.

## 2 The PRank Algorithm

This paper focuses on online algorithms for ranking instances. We are given a sequence $(\mathbf{x}^1, y^1), \ldots, (\mathbf{x}^t, y^t), \ldots$ of instance-rank pairs. Each instance $\mathbf{x}^t$ is in $\mathbb{R}^n$ and its corresponding rank $y^t$ is an element from finite set $\mathcal{Y}$ with a total order relation. We assume without loss of generality that $\mathcal{Y} = \{1, 2, \ldots, k\}$ with ">" as the order relation. The total order over the set $\mathcal{Y}$ induces a partial order over the instances in the following natural sense. We say that $\mathbf{x}^t$ is preferred over $\mathbf{x}^s$ if $y^t > y^s$. We also say that $\mathbf{x}^t$ and $\mathbf{x}^s$ are not comparable if neither $y^t > y^s$ nor $y^t < y^s$. We denote this case simply as $y^t = y^s$. Note that the induced partial order is of a unique form in which the instances form $k$ equivalence classes which are totally ordered[1]. A *ranking rule* $H$ is a mapping from instances to ranks, $H : \mathbb{R}^n \to \mathcal{Y}$. The family of ranking rules we discuss in this paper employs a vector $\mathbf{w} \in \mathbb{R}^n$ and a set of $k$ thresholds $b_1 \leq \ldots \leq b_{k-1} \leq b_k = \infty$. For convenience we denote by $\mathbf{b} = (b_1, \ldots, b_{k-1})$ the vector of thresholds excluding $b_k$ which is fixed to $\infty$. Given a new instance $\mathbf{x}$ the ranking rule first computes the inner-product between $\mathbf{w}$ and $\mathbf{x}$. The predicted rank is then defined to be the index of the first (smallest) threshold $b_r$ for which $\mathbf{w} \cdot \mathbf{x} < b_r$. This type of ranking rules divide the space into parallel equally-ranked regions: all the instances that satisfy $b_{r-1} < \mathbf{w} \cdot \mathbf{x} < b_r$ are assigned the same rank $r$. Formally, given a ranking rule defined by $\mathbf{w}$ and $\mathbf{b}$ the predicted rank of an instance $\mathbf{x}$ is, $H(\mathbf{x}) = \min_{r \in \{1, \ldots, k\}} \{r : \mathbf{w} \cdot \mathbf{x} - b_r < 0\}$. Note that the above minimum is always well defined since we set $b_k = \infty$.

The analysis that we use in this paper is based on the mistake bound model for online learning. The algorithm we describe works in rounds. On round $t$ the learning algorithm gets an instance $\mathbf{x}^t$. Given $\mathbf{x}^t$, the algorithm outputs a rank, $\hat{y}^t = \min_r \{r : \mathbf{w} \cdot \mathbf{x} - b_r < 0\}$. It then receives the correct rank $y^t$ and updates its ranking rule by modifying $\mathbf{w}$ and $\mathbf{b}$. We say that our algorithm made a ranking mistake if $\hat{y}^t \neq y^t$.

**Initialize:** Set $\mathbf{w}^1 = 0$ , $b_1^1, \ldots, b_{k-1}^1 = 0, b_k^1 = \infty$ .
**Loop:** For $t = 1, 2, \ldots, T$

- Get a new rank-value $\mathbf{x}^t \in \mathbb{R}^n$.
- Predict $\hat{y}^t = \min_{r \in \{1, \ldots, k\}} \{r : \mathbf{w}^t \cdot \mathbf{x}^t - b_r^t < 0\}$.
- Get a new label $y^t$.
- If $\hat{y}^t \neq y^t$ update $\mathbf{w}^t$ (otherwise set $\mathbf{w}^{t+1} = \mathbf{w}^t$ , $\forall r : b_r^{t+1} = b_r^t$) :

  1. For $r = 1, \ldots, k-1$ :  If $y^t \leq r$ Then $y_r^t = -1$
     Else $y_r^t = 1$.
  2. For $r = 1, \ldots, k-1$ :  If $(\mathbf{w}^t \cdot \mathbf{x}^t - b_r^t)y_r^t \leq 0$ Then $\tau_r^t = y_r^t$
     Else $\tau_r^t = 0$.
  3. Update $\mathbf{w}^{t+1} \leftarrow \mathbf{w}^t + (\sum_r \tau_r^t)\mathbf{x}^t$.
     For $r = 1, \ldots, k-1$ update:    $b_r^{t+1} \leftarrow b_r^t - \tau_r^t$

**Output :**   $H(\mathbf{x}) = \min_{r \in \{1, \ldots, k\}} \{r : \mathbf{w}^{T+1} \cdot \mathbf{x} - b_r^{T+1} < 0\}$.

Figure 2: The PRank algorithm.

We wish to make the predicted rank as close as possible to the true rank. Formally, the goal of the learning algorithm is to minimize the ranking-loss which is defined to be the number of thresholds between the true rank and the predicted rank. Using the representation of ranks as integers in $\{1 \ldots k\}$, the ranking-loss after $T$ rounds is equal to the accumulated difference between the predicted and true rank-values, $\sum_{t=1}^{T} |\hat{y}^t - y^t|$. The algorithm we describe updates its ranking rule only on rounds on which it made ranking mistakes. Such algorithms are called *conservative*.

We now describe the update rule of the algorithm which is motivated by the perceptron algorithm for classification and hence we call it the PRank algorithm (for Perceptron Ranking). For simplicity, we omit the index of the round when referring to an input instance-rank pair $(\mathbf{x}, y)$ and the ranking rule $\mathbf{w}$ and $\mathbf{b}$. Since $b_1 \leq b_2 \leq \ldots \leq b_{k-1} \leq b_k$ then the predicted rank is correct if $\mathbf{w} \cdot \mathbf{x} > b_r$ for $r = 1, \ldots, y-1$ and $\mathbf{w} \cdot \mathbf{x} < b_r$ for $y, \ldots, k-1$. We represent the above inequalities by expanding the rank $y$ into into $k-1$ virtual variables $y_1, \ldots, y_{k-1}$. We set $y_r = +1$ for the case $\mathbf{w} \cdot \mathbf{x} > b_r$ and $y_r = -1$ for $\mathbf{w} \cdot \mathbf{x} < b_r$. Put another way, a rank value $y$ induces the vector $(y_1, \ldots, y_{k-1}) = (+1, \ldots, +1, -1, \ldots, -1)$ where the maximal index $r$ for which $y_r = +1$ is $y-1$. Thus, the prediction of a ranking rule is correct if $y_r(\mathbf{w} \cdot \mathbf{x} - b_r) > 0$ for all $r$. If the algorithm makes a mistake by ranking $\mathbf{x}$ as $\hat{y}$ instead of $y$ then there is at least one threshold, indexed $r$, for which the value of $\mathbf{w} \cdot \mathbf{x}$ is on the wrong side of $b_r$, i.e. $y_r(\mathbf{w} \cdot \mathbf{x} - b_r) \leq 0$. To correct the mistake, we need to "move" the values of $\mathbf{w} \cdot \mathbf{x}$ and $b_r$ toward each other. We do so by modifying only the values of the $b_r$'s for which $y_r(\mathbf{w} \cdot \mathbf{x} - b_r) \leq 0$ and replace them with $b_r - y_r$. We also replace the value of $\mathbf{w}$ with $\mathbf{w} + (\sum y_r)\mathbf{x}$ where the sum is taken over the indices $r$ for which there was a prediction error, i.e., $y_r(\mathbf{w} \cdot \mathbf{x} - b_r) \leq 0$.

An illustration of the update rule is given in Fig 1. In the example, we used the set $\mathcal{Y} = \{1 \ldots 5\}$. (Note that $b_5 = \infty$ is omitted from all the plots in Fig 1.) The correct rank of the instance is $y = 4$, and thus the value of $\mathbf{w} \cdot \mathbf{x}$ should fall in the fourth interval, between $b_3$ and $b_4$. However, in the illustration the value of $\mathbf{w} \cdot \mathbf{x}$ fell below $b_1$ and the predicted rank is $\hat{y} = 1$. The threshold values $b_1$, $b_2$ and $b_3$ are a source of the error since the value of $b_1$, $b_2$, $b_3$ is higher then $\mathbf{w} \cdot \mathbf{x}$. To mend the mistake the algorithm decreases $b_1$, $b_2$ and $b_3$ by a unit value and replace them with $b_1 - 1$, $b_2 - 1$ and $b_3 - 1$. It also modifies $\mathbf{w}$ to be $\mathbf{w} + 3\mathbf{x}$ since $\sum_{r:y_r(\mathbf{w} \cdot \mathbf{x} - b_r) \leq 0} y_r = 3$. Thus, the inner-product $\mathbf{w} \cdot \mathbf{x}$ increases by $3\|\mathbf{x}\|^2$. This update is illustrated at the middle plot of Fig. 1. The updated prediction rule is sketched on the right hand

side of Fig. 1. Note that after the update, the predicted rank of $\mathbf{x}$ is $\hat{y} = 3$ which is closer to the true rank $y = 4$. The pseudocode of algorithm is given in Fig 2.

To conclude this section we like to note that PRank can be straightforwardly combined with Mercer kernels [8] and voting techniques [4] often used for improving the performance of margin classifiers in batch and online settings.

## 3   Analysis

Before we prove the mistake bound of the algorithm we first show that it maintains a consistent hypothesis in the sense that it preserves the correct order of the thresholds. Specifically, we show by induction that for any ranking rule that can be derived by the algorithm along its run, $(\mathbf{w}^1, \mathbf{b}^1)\,,\ldots\,,(\mathbf{w}^{T+1}, \mathbf{b}^{T+1})$ we have that $b_r^t \leq \ldots \leq b_{k-1}^t$ for all $t$. Since the initialization of the thresholds is such that $b_1^1 \leq b_2^1 \leq \ldots \leq b_{k-1}^1$, then it suffices to show that the claim holds inductively. For simplicity, we write the updating rule of PRank in an alternative form. Let $\llbracket \pi \rrbracket$ be 1 if the predicate $\pi$ holds and 0 otherwise. We now rewrite the value of $\tau_r^t$ (from Fig. 2) as $\tau_r^t = y_r^t \llbracket (\mathbf{w}^t \cdot \mathbf{x}^t - b_r^t) y_r^t \leq 0 \rrbracket$. Note that the values of $b_r^t$ are integers for all $r$ and $t$ since for all $r$ we initialize $b_r^1 = 0$, and $b_r^{t+1} - b_r^t \in \{-1, 0, +1\}$.

**Lemma 1 (Order Preservation)** *Let $\mathbf{w}^t$ and $\mathbf{b}^t$ be the current ranking rule, where $b_1^t \leq \ldots \leq b_{k-1}^t$, and let $(\mathbf{x}^t, y^t)$ be an instance-rank pair fed to PRank on round $t$. Denote by $\mathbf{w}^{t+1}$ and $\mathbf{b}^{t+1}$ the resulting ranking rule after the update of PRank, then $b_1^{t+1} \leq \ldots \leq b_{k-1}^{t+1}$.*

**Proof:**   In order to show that PRank maintains the order of the thresholds we use the definition of the algorithm for $y_r^t$, namely we define $y_r^t = +1$ for $r < y^t$ and $y_r^t = -1$ for $r \geq y^t$. We now prove that $b_{r+1}^{t+1} \geq b_r^{t+1}$ for all $r$ by showing that

$$b_{r+1}^t - b_r^t \geq y_{r+1}^t \llbracket (\mathbf{w}^t \cdot \mathbf{x}^t - b_{r+1}^t) y_{r+1}^t \leq 0 \rrbracket - y_r^t \llbracket (\mathbf{w}^t \cdot \mathbf{x}^t - b_r^t) y_r^t \leq 0 \rrbracket \, , \quad (1)$$

which we obtain by substituting the values of $\mathbf{b}^{t+1}$. Since $b_{r+1}^t \leq b_r^t$ and $b_r^t, b_{r+1}^t \in \mathbb{Z}$ we get that the value of $b_{r+1}^t - b_r^t$ on the left hand side of Eq. (1) is a non-negative integer. Recall that $y_r^t = 1$ if $y^t > r$ and $y_r^t = -1$ otherwise, and therefore, $y_{r+1}^t \leq y_r^t$. We now analyze two cases. We first consider the case $y_{r+1}^t \neq y_r^t$ which implies that $y_{r+1}^t = -1$, $y_r^t = +1$. In this case, the right hand-side of Eq. (1) is at most zero, and the claim trivially holds. The other case is when $y_{r+1}^t = y_r^t$. Here we get that the value of the right hand-side Eq. (1) cannot exceed 1. We therefore have to consider only the case where $b_r^t = b_{r+1}^t$ and $y_{r+1}^t = y_r^t$. But given these two conditions we have that $y_{r+1}^t \llbracket (\mathbf{w}^t \cdot \mathbf{x}^t - b_{r+1}^t) y_{r+1}^t < 0 \rrbracket$ and $y_r^t \llbracket (\mathbf{w}^t \cdot \mathbf{x}^t - b_r^t) y_r^t < 0 \rrbracket$ are equal. The right hand side of Eq. (1) is now zero and the inequality holds with equality. ∎

In order to simplify the analysis of the algorithm we introduce the following notation. Given a hyperplane $\mathbf{w}$ and a set of $k-1$ thresholds $\mathbf{b}$ we denote by $\mathbf{v} \in \mathbb{R}^{n+k-1}$ the vector which is a concatenation of $\mathbf{w}$ and $\mathbf{b}$ that is $\mathbf{v} = (\mathbf{w}, \mathbf{b})$. For brevity we refer to the vector $\mathbf{v}$ as a *ranking rule*. Given two vectors $\mathbf{v}' = (\mathbf{w}', \mathbf{b}')$ and $\mathbf{v} = (\mathbf{w}, \mathbf{b})$ we have $\mathbf{v}' \cdot \mathbf{v} = \mathbf{w}' \cdot \mathbf{w} + \mathbf{b}' \cdot \mathbf{b}$ and $\|\mathbf{v}\|^2 = \|\mathbf{w}\|^2 + \|\mathbf{b}\|^2$.

**Theorem 2 (Mistake bound)** *Let $(\mathbf{x}^1, y^1), \ldots, (\mathbf{x}^T, y^T)$ be an input sequence for PRank where $\mathbf{x}^t \in \mathbb{R}^n$ and $y^t \in \{1 \ldots k\}$. Denote by $R^2 = \max_t \|\mathbf{x}^t\|^2$. Assume that there is a ranking rule $\mathbf{v}^* = (\mathbf{w}^*, \mathbf{b}^*)$ with $b_1^* \leq \ldots \leq b_{k-1}^*$ of a unit norm that classifies the entire sequence correctly with margin $\gamma = \min_{r,t}\{(\mathbf{w}^* \cdot \mathbf{x}^t - b_r^*) y_r^t\} > 0$. Then, the rank loss of the algorithm $\sum_{t=1}^T |\hat{y}^t - y^t|$, is at most $(k-1)(R^2+1)/\gamma^2$.*

**Proof:** Let us fix an example $(\mathbf{x}^t, y^t)$ which the algorithm received on round $t$. By definition the algorithm ranked the example using the ranking rule $\mathbf{v}^t$ which is composed of $\mathbf{w}^t$ and the thresholds $\mathbf{b}^t$. Similarly, we denote by $\mathbf{v}^{t+1}$ the updated rule $(\mathbf{w}^{t+1}, \mathbf{b}^{t+1})$ after round $t$. That is, $\mathbf{w}^{t+1} = \mathbf{w}^t + (\sum_r \tau_r^t)\mathbf{x}^t$ and $b_r^{t+1} = b_r^t - \tau_r^t$ for $r = 1, 2, \ldots, k-1$. Let us denote by $n^t = |\hat{y}^t - y^t|$ the difference between the true rank and the predicted rank. It is straightforward to verify that $n^t = \sum_r |\tau_r^t|$. Note that if there wasn't a ranking mistake on round $t$ then $\tau_r^t = 0$ for $r = 1, \ldots, k-1$, and thus also $n^t = 0$. To prove the theorem we bound $\sum_t n^t$ from above by bounding $\|\mathbf{v}^t\|^2$ from above and below. First, we derive a lower bound on $\|\mathbf{v}^t\|^2$ by bounding $\mathbf{v}^* \cdot \mathbf{v}^{t+1}$. Substituting the values of $\mathbf{w}^{t+1}$ and $\mathbf{b}^{t+1}$ we get,

$$\mathbf{v}^* \cdot \mathbf{v}^{t+1} = \mathbf{v}^* \cdot \mathbf{v}^t + \sum_{r=1}^{k-1} \tau_r^t \left(\mathbf{v}^* \cdot \mathbf{x}^t - b_r^*\right) \tag{2}$$

We further bound the right term by considering two cases. Using the definition of $\tau_r^t$ from the pseudocode in Fig. 2 we need to analyze two cases. If $(\mathbf{w}^t \cdot \mathbf{x}^t - b_r^t)y_r^t \leq 0$ then $\tau_r^t = y_r^t$. Using the assumption that $\mathbf{v}^*$ ranks the data correctly with a margin of at least $\gamma$ we get that $\tau_r^t(\mathbf{w}^* \cdot \mathbf{x}^t - b_r^*) \geq \gamma$. For the other case for which $(\mathbf{w}^t \cdot \mathbf{x}^t - b_r^t)y_r^t > 0$ we have $\tau_r^t = 0$ and thus $\tau_r^t(\mathbf{w}^* \cdot \mathbf{x}^t - b_r^*) = 0$. Summing now over $r$ we get,

$$\sum_{r=1}^{k-1} \tau_r^t \left(\mathbf{w}^* \cdot \mathbf{x}^t - b_r^*\right) \geq n^t \gamma . \tag{3}$$

Combining Eq. (2) and Eq. (3) we get $\mathbf{v}^* \cdot \mathbf{v}^{t+1} \geq \mathbf{v}^* \cdot \mathbf{v}^t + n^t \gamma$. Unfolding the sum, we get that after $T$ rounds the algorithm satisfies, $\mathbf{v}^* \cdot \mathbf{v}^{T+1} \geq \sum_t n^t \gamma = \gamma \sum_t n^t$. Plugging this result into Cauchy-Schwartz inequality, $(\|\mathbf{v}^{T+1}\|^2\|\mathbf{v}^*\|^2 \geq (\mathbf{v}^{T+1} \cdot \mathbf{v}^*)^2)$ and using the assumption that $\mathbf{v}^*$ is of a unit norm we get the lower bound, $\|\mathbf{v}^{T+1}\|^2 \geq (\sum_t n^t)^2 \gamma^2$.

Next, we bound the norm of $\mathbf{v}$ from above. As before, assume that an example $(\mathbf{x}^t, y^t)$ was ranked using the ranking rule $\mathbf{v}^t$ and denote by $\mathbf{v}^{t+1}$ the ranking rule after the round. We now expand the values of $\mathbf{w}^{t+1}$ and $\mathbf{b}^{t+1}$ in the norm of $\mathbf{v}^{t+1}$ and get, $\|\mathbf{v}^{t+1}\|^2 = \|\mathbf{w}^t\|^2 + \|\mathbf{b}^t\|^2 + 2\sum_r \tau_r^t(\mathbf{w}^t \cdot \mathbf{x}^t - b_r^t) + (\sum_r \tau_r^t)^2\|\mathbf{x}^t\|^2 + \sum_r (\tau_r^t)^2$. Since $\tau_r^t \in \{-1, 0, +1\}$ we have that $(\sum_r \tau_r^t)^2 \leq (n^t)^2$ and $\sum_r (\tau_r^t)^2 = n^t$ and we therefore get,

$$\|\mathbf{v}^{t+1}\|^2 \leq \|\mathbf{v}^t\|^2 + 2\sum_r \tau_r^t \left(\mathbf{w}^t \cdot \mathbf{x}^t - b_r^t\right) + (n^t)^2\|\mathbf{x}^t\|^2 + n^t . \tag{4}$$

We further develop the second term using the update rule of the algorithm and get,

$$\sum_r \tau_r^t \left(\mathbf{w}^t \cdot \mathbf{x}^t - b_r^t\right) = \sum_r [\![(\mathbf{w}^t \cdot \mathbf{x}^t - b_r^t)y_r^t \leq 0]\!] \left((\mathbf{w}^t \cdot \mathbf{x}^t - b_r^t)y_r^t\right) \leq 0 . \tag{5}$$

Plugging Eq. (5) into Eq. (4) and using the bound $\|\mathbf{x}^t\|^2 \leq R^2$ we get that $\|\mathbf{v}^{t+1}\|^2 \leq \|\mathbf{v}^t\|^2 + (n^t)^2 R^2 + n^t$. Thus, the ranking rule we obtain after $T$ rounds of the algorithm satisfies the upper bound, $\|\mathbf{v}^{T+1}\|^2 \leq R^2 \sum_t (n^t)^2 + \sum_t n^t$. Combining the lower bound $\|\mathbf{v}^{T+1}\|^2 \geq (\sum_t n^t)^2 \gamma^2$ with the upper bound we have that, $(\sum_t n^t)^2 \gamma^2 \leq \|\mathbf{v}^{T+1}\|^2 \leq R^2 \sum_t (n^t)^2 + \sum_t n^t$. Dividing both sides by $\gamma^2 \sum_t n^t$ we finally get,

$$\sum_t n^t \leq \frac{R^2 \left[\sum_t (n^t)^2\right] / \left[\sum_t n^t\right] + 1}{\gamma^2} . \tag{6}$$

By definition, $n^t$ is at most $k-1$, which implies that $\sum_t (n^t)^2 \leq \sum_t n^t(k-1) = (k-1)\sum_t n^t$. Using this inequality in Eq. (6) we get the desired bound, $\sum_{t=1}^T |\hat{y}^t - y^t| = \sum_{t=1}^T n^t \leq [(k-1)R^2 + 1]/\gamma^2 \leq [(k-1)(R^2+1)]/\gamma^2 .$ ∎

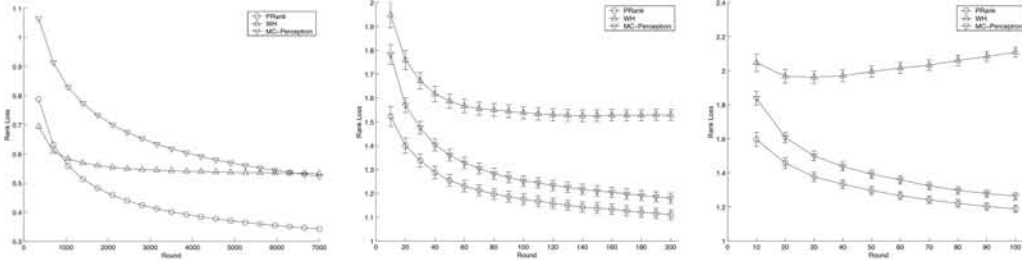

Figure 3: Comparison of the time-averaged ranking-loss of PRank, WH, and MCP on synthetic data (left). Comparison of the time-averaged ranking-loss of PRank, WH, and MCP on the `EachMovie` dataset using viewers who rated and at least 200 movies (middle) and at least 100 movies (right).

## 4  Experiments

In this section we describe experiments we performed that compared PRank with two other online learning algorithms applied to ranking: a multiclass generalization of the perceptron algorithm [2], denoted MCP, and the Widrow-Hoff [9] algorithm for online regression learning which we denote by WH. For WH we fixed its learning rate to a constant value. The hypotheses the three algorithms maintain share similarities but are different in their complexity: PRank maintains a vector $\mathbf{w}$ of dimension $n$ and a vector of $k-1$ modifiable thresholds $\mathbf{b}$, totaling $n+k-1$ parameters; MCP maintains $k$ prototypes which are vectors of size $n$, yielding $kn$ parameters; WH maintains a single vector $\mathbf{w}$ of size $n$. Therefore, MCP builds the most complex hypothesis of the three while WH builds the simplest.

Due to the lack of space, we only describe two sets of experiments with two different datasets. The dataset used in the first experiment is synthetic and was generated in a similar way to the dataset used by Herbrich et. al. [5]. We first generated random points $\mathbf{x} = (x_1, x_2)$ uniformly at random from the unit square $[0,1]^2$. Each point was assigned a rank $y$ from the set $\{1, \ldots, 5\}$ according to the following ranking rule, $y = \max_r \{r : 10((x_1 - 0.5)(x_2 - 0.5)) + \xi > b_r\}$ where $\mathbf{b} = (-\infty, -1, -0.1, 0.25, 1)$ and $\xi$ is a normally distributed noise of a zero mean and a standard deviation of 0.125. We generated 100 sequences of instance-rank pairs each of length 7000. We fed the sequences to the three algorithms and obtained a prediction for each instance. We converted the real-valued predictions of WH into ranks by rounding each prediction to its closest rank value. As in [5] we used a non-homogeneous polynomial of degree 2, $K(\mathbf{x}_1, \mathbf{x}_2) = ((\mathbf{x}_1 \cdot \mathbf{x}_2) + 1)^2$ as the inner-product operation between each input instance and the hyperplanes the three algorithms maintain. At each time step, we computed for each algorithm the accumulated ranking-loss normalized by the instantaneous sequence length. Formally, the time-averaged loss after $T$ rounds is, $(1/T) \sum_t^T |\hat{y}^t - y^t|$. We computed these losses for $T = 1, \ldots, 7000$. To increase the statistical significance of the results we repeated the process 100 times, picking a new random instance-rank sequence of length $7,000$ each time, and averaging the instantaneous losses across the 100 runs. The results are depicted on the left hand side of Fig. 3. The 95% confidence intervals are smaller then the symbols used in the plot. In this experiment the performance of MPC is constantly worse than the performance of WH and PRank. WH initially suffers the smallest instantaneous loss but after about 500 rounds PRank achieves the best performance and eventually the number of ranking mistakes that PRank suffers is significantly lower than both WH and MPC.

In the second set of experiments we used the `EachMovie` dataset [7]. This dataset is used for collaborative filtering tasks and contains ratings of movies provided by $61,265$ people. Each person in the dataset viewed a subset of movies from a collection of 1623 titles. Each viewer rated each movie that she saw using one of 6 possible ratings: $0, 0.2, 0.4, 0.6, 0.8, 1$. We chose subsets of people who viewed a significant amount of movies extracting for evaluation people who have rated at least 100 movies. There were $7,542$ such viewers. We chose at random one person among these viewers and set the person's ratings to be the target rank. We used the ratings of all the rest of the people who viewed enough movies as features. Thus, the goal is to learn to predict the "taste" of a random user using the user's past ratings as a feedback and the ratings of fellow viewers as features. The prediction rule associates a weight with each fellow viewer an therefore can be seen as learning correlations between the tastes of different viewers. Next, we subtracted 0.5 from each rating and therefore the possible ratings are $-0.5, -0.3, -0.1, 0.1, 0.3, 0.5$. This linear transformation enabled us to assign a value of zero to movies which have not been rated. We fed these feature-rank pairs one at a time, in an online fashion. Since we picked viewer who rated at least 100 movies, we were able to perform at least 100 rounds of online predictions and updates. We repeated this experiment 500 times, choosing each time a random viewer for the target rank. The results are shown on the right hand-side of Fig. 3. The error bars in the plot indicate 95% condfidence levels. We repeated the experiment using viewers who have seen at least 200 movies. (There were 1802 such viewers.) The results of this experiment are shown in the middle plot of Fig. 3. Along the entire run of the algorithms, PRank is significantly better than WH, and consistently better than the multiclass perceptron algorithm, although the latter employs a bigger hypothesis.

Finally, we have also evaluated the performance of PRank in a batch setting, using the experimental setup of [5]. In this experiment, we ran PRank over the training data as an online algorithm and used its last hypothesis to rank unseen test data. Here as well PRank came out first, outperforming all the algorithms described in [5].

**Acknowledgments** Thanks to Sanjoy Dagupta and Rob Schapire for numerous discussions on ranking problems and algorithms. Thanks also to Eleazar Eskin and Uri Maoz for carefully reading the manuscript.

## Footnotes

[1]For a discussion of this type of partial orders see [6].

# References

[1] William W. Cohen, Robert E. Schapire, and Yoram Singer. Learning to order things. *Journal of Artificial Intelligence Research*, 10:243–270, 1999.

[2] K. Crammer and Y. Singer. Ultraconservative online algorithms for multiclass problems. *Proc. of the Fourteenth Annual Conf. on Computational Learning Theory*, 2001.

[3] Y. Freund, R. Iyer, R. E. Schapire, and Y. Singer. An efficient boosting algorithm for combining preferences. *Machine Learning: Proc. of the Fifteenth Intl. Conf.*, 1998.

[4] Y. Freund and R. E. Schapire. Large margin classification using the perceptron algorithm. *Machine Learning*, 37(3): 277-296, 1999.

[5] R. Herbrich, T. Graepel, and K. Obermayer. Large margin rank boundaries for ordinal regression. *Advances in Large Margin Classifiers*. MIT Press, 2000.

[6] J. Kemeny and J. Snell. *Mathematical Models in the Social Sciences*. MIT Press, 1962.

[7] Paul McJones. EachMovie collaborative filtering data set. DEC Systems Research Center, 1997. http://www.research.digital.com/SRC/eachmovie/.

[8] Vladimir N. Vapnik. *Statistical Learning Theory*. Wiley, 1998.

[9] Bernard Widrow and Marcian E. Hoff. Adaptive switching circuits. *1960 IRE WESCON Convention Record*, 1960. Reprinted in *Neurocomputing* (MIT Press, 1988).
